# Non-linear Statistical Analysis and Self-Organizing Hebbian Networks

**Jonathan L. Shapiro and Adam Prügel-Bennett**
Department of Computer Science
The University, Manchester
Manchester, UK
M13 9PL

## Abstract

Neurons learning under an unsupervised Hebbian learning rule can perform a nonlinear generalization of principal component analysis. This relationship between nonlinear PCA and nonlinear neurons is reviewed. The stable fixed points of the neuron learning dynamics correspond to the maxima of the statistic optimized under nonlinear PCA. However, in order to predict what the neuron learns, knowledge of the basins of attractions of the neuron dynamics is required. Here the correspondence between nonlinear PCA and neural networks breaks down. This is shown for a simple model. Methods of statistical mechanics can be used to find the optima of the objective function of non-linear PCA. This determines what the neurons can learn. In order to find how the solutions are partitioned amoung the neurons, however, one must solve the dynamics.

## 1 INTRODUCTION

Linear neurons learning under an unsupervised Hebbian rule can learn to perform a linear statistical analysis of the input data. This was first shown by Oja (1982), who proposed a learning rule which finds the first principal component of the variance matrix of the input data. Based on this model, Oja (1989), Sanger (1989), and many others have devised numerous neural networks which find many components of this matrix. These networks perform principal component analysis (PCA), a well-known method of statistical analysis.

Since PCA is a form of linear analysis, and the neurons used in the PCA networks are linear – the output of these neurons is equal to the weighted sum of inputs; there is no squashing function of sigmoid – it is obvious to ask whether non-linear Hebbian neurons compute some form of non-linear PCA? Is this a useful way to understand the performance of the networks? Do these networks learn to extract features of the input data which are different from those learned by linear neurons? Currently in the literature, the phrase "non-linear PCA" is used to describe what is learned by any non-linear generalization of Oja neurons or other PCA networks (see for example, Oja, 1993 and Taylor, 1993).

In this paper, we discuss the relationship between a particular form of non-linear Hebbian neurons (Prügel-Bennett and Shapiro, 1992) and a particular generalization of non-linear PCA (Softky and Kammen 1991). It is clear that non-linear neurons can perform very differently from linear ones. This has been shown through analysis (Prügel-Bennett and Shapiro, 1993) and in application (Karhuenen and Joutsensalo, 1992). It can also be very useful way of understanding what the neurons learn. This is because non-linear PCA is equivalent to maximizing some objective function. The features that this extracts from a data set can be studied using techniques of statistical mechanics. However, non-linear PCA is ambiguous because there are multiple solutions. What the neuron *can* learn is given by non-linear PCA. The likelihood of learning the different solutions is governed by the dyanamics chosen to implement non-linear PCA, and may differ in different implementations of the dynamics.

## 2   NON-LINEAR HEBBIAN NEURONS

Neurons with non-linear activation functions can learn to perform very different tasks from those learned by linear neurons. Nonlinear Hebbian neurons have been analyzed for general non-linearities by Oja (1991), and was applied to sinusoidal signal detection by Karhuenen and Joutsensalo (1992).

Previously, we analysed a simple non-linear generalization of Oja's rule (Prügel-Bennett and Shapiro, 1993). We showed how the shape of the neuron activation function can control what a neuron learns. Whereas linear neurons learn to a statistic mixture of all of the input patterns, non-linear neurons can learn to become tuned to individual patterns, or to small clusters of closely correlated patterns.

In this model, each neuron has weights, $w_i$ is the weight from the $i^{th}$ input, and responds to the usual sum of input times weights through an activation function $A(y)$. This is assumed a simple power-law above a threshold and zero below it. I.e.

$$A(V^p) = \begin{cases} (V^p - \phi)^b & V^p > \phi \\ 0 & V^p \leq \phi \end{cases} \tag{1}$$

Here $\phi$ is the threshold, $b$ controls the power of the power-law, $x_i^p$ is the ith component of the pth pattern, and $V^p = \sum_i x_i^p w_i$. Curves of these functions are shown in figure 1a; if $b = 1$ the neurons are threshold-linear. For $b > 1$ the curves can be thought of as low activation approximations to a sigmoid which is shown in figure 1b. The generalization of Oja's learning rule is that the change in the weights $\delta w_i$

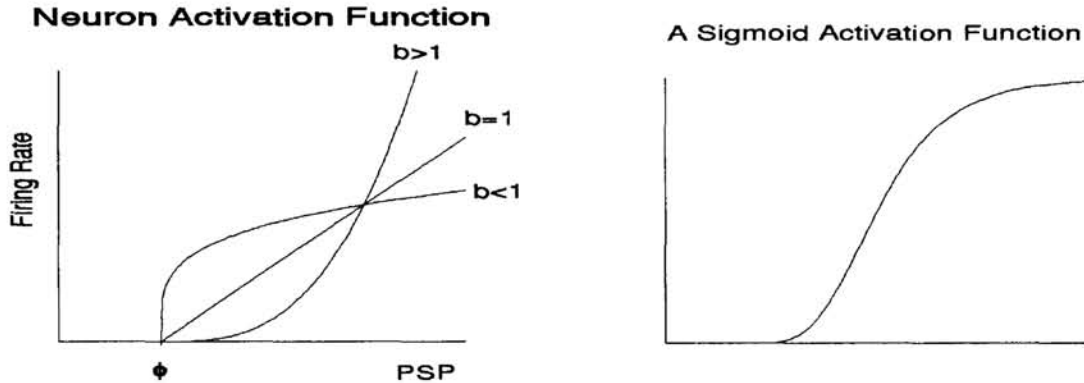

Figure 1: a) The form of the neuron activation function. Control by two parameters $b$ and $\phi$. When $b > 1$, this activation function approximates a sigmoid, which is shown in b).

is given by

$$\delta w_i = \sum_p A(V^p) \left[ x_i^p - V^p w_i \right].  \tag{2}$$

If $b < 1$, the neuron learns to average a set of patterns. If $b = 1$, the neuron finds the principal component of the pattern set. When $b > 1$, the neuron learns to distinguish one of the patterns in the presence of the others, if those others are not too correlated with the pattern. There is a critical correlation which is determined by $b$; the neuron learns to individual patterns which are less correlated than the critical value, but learns to something like the center of the cluster if the patterns are more correlated. The threshold controls the size of the subset of patterns which the neuron can respond to.

For these neurons, the relationship between non-PCA and the activation function was not previously discussed. That is done in the next section.

## 3    NON-LINEAR PCA

A non-linear generalization of PCA was proposed by Softky and Kammen (1991). In this section, the relationship between non-linear PCA and unsupervised Hebbian learning is reviewed.

## 3.1   WHAT IS NON-LINEAR PCA

The principal component of a set of data is the direction which maximises the variance. I.e. to find the principal component of the data set, find the vector $\vec{w}$ of unit length which maximises

$$H = \left\langle \left(\sum_i w_i x_i\right)^2 \right\rangle. \tag{3}$$

Here, $x_i$ denotes the $i^{\text{th}}$ component of an input pattern and $< \cdots >$ denotes the average over the patterns. Sofky and Kammen suggested that an appropriate generalization is to find the vector $\vec{w}$ which maximizes the *d-dimensional correlation*,

$$H = \left\langle \left|\sum_i w_i x_i\right|^d \right\rangle. \tag{4}$$

They argued this would give interesting results if higher order correlations are important, or if the shape of the data cloud is not second order. This can be generalized further, of course, maximizing the average of any non-linear function of the input $U(y)$,

$$H = < U\left(\sum_i w_i x_i\right) > . \tag{5}$$

The equations for the principal components are easily found using Lagrange multipliers. The extremal points are given by

$$< U'\left(\sum_k w_k x_k\right) x_i > = \lambda w_i. \tag{6}$$

These points will be (local) maxima if the Hessian $\mathcal{H}_{ij}$,

$$\mathcal{H}_{ij} = < U''\left(\sum_k w_k x_k\right) x_i x_j > - \lambda \delta_{ij}, \tag{7}$$

Here, $\lambda$ is a Lagrange multiplier chosen to make $|\vec{w}|^2 = 1$.

## 3.2   NEURONS WHICH LEARN PCA

A neuron learning via unsupervised Hebbian learning rule can perform this optimization. This is done by associating $w_i$ with the weight from the ith input to the neuron, and the data average $< \cdot >$ as the sum over input patterns $x_i^p$. The nonlinear function which is optimized is determined by the integral of the activation function of the neuron

$$A(y) = U'(y).$$

In their paper, Softky and Kammen propose a learning rule which does not perform this optimization in general. The correct learning rule is a generalization of Oja's rule (equation (2) above), in this notation,

$$\delta w_i = \langle A(V)\,[x_i - V w_i] \rangle. \tag{8}$$

This fixed points of this dynamical equation will be solutions to the extremal equation of nonlinear PCA, equation (6), when the associations

$$\lambda = \langle A(V)V \rangle,$$

and

$$A(y) = U'(y)$$

are made.

Here $\langle \cdot \rangle$ is interpreted as sum over patterns; this is batch learning. The rule can also be used incrementally, but then the dynamics are stochastic and the optimization might be performed only on average, and then maybe only for small enough learning rates. These fixed points will be stable when the Hessian $\mathcal{H}_{ij}$ is negative definite at the fixed point. This is now,

$$\mathcal{H}_{ij} = <U''(V)x_i x_j> - \lambda \delta_{ij} - w_i w_j \lambda - <VU''(V)x_j> w_i. \tag{9}$$

which is the same as the previous, equation (7),in directions perpendicular to the fixed point, but contains additional terms in direction of the fixed point which normalize it.

The neurons described in section 2 would perform precisely what Softky and Kammen proposed if the activation function was pure power-law and not thresholded; as it is they maximize a more complicated objective function.

Since there is a one to one correspondence between the stable fixed points of the dynamics and the *local* maxima of the non-linear correlation measure, one says that these non-linear neurons compute non-linear PCA.

## 3.3   THEORETICAL STUDIES OF NONLINEAR PCA

In order to understand what these neurons learn, we have studied the networks learning on model data drawn from statistical distributions. For very dense clusters $P \to \infty, N$ fixed, the stable fixed point equations are algebraic. In a few simple cases they can be solved. For example, if the data is Gaussian or if the data cloud is a quadratic cloud (a function of a quadratic form), the neuron learns the principal component, like the linear neuron. Likewise, if the patterns are not random, the fixed point equations can be solved in some cases.

For large number of patterns in high dimensions fluctuations in the data are important ($N$ and $P$ goes to $\infty$ together in some way). In this case, methods of statistical mechanics can be used to average over the data. The objective function of the non-linear PCA acts as (minus) the energy in statistical mechanics. The free energy is formally,

$$F = <\log(\prod_{i=1}^{N} \int w_i \, \delta \left( \sum_{i=1}^{N} w_i^2 - 1 \right) \exp \beta U(V) > . \tag{10}$$

In the limit that $\beta$ is large, this calculation finds the local maxima of $U$. In this form of analysis, the fact that the neuron optimizes an objective function is very important. This technique was used to produce the results outlined in section 2.

### 3.4    WHAT NON-LINEAR PCA FAILS TO REVEAL

In the linear PCA, there is one unique solution, or if there are many solutions it is because the solutions are degenerate. However, for the non-linear situation, there are many stable fixed points of the dynamics and many local maxima of the non-linear correlation measure.

This has two effects. First, it means that you cannot predict what the neuron will learn simply by studying fixed point equations. This tells you what the neuron might learn, but the probability that this solution will be can only be ascertained if the dynamics are understood. This also breaks the relationship between non-linear PCA and the neurons, because, in principle, there could be other dynamics which have the same fixed point structure, but do not have the same basins of attraction. Simple fixed point analysis would be incapable of predicting what these neurons would learn.

## 4    PARTITIONING

An important question which the fixed-point analysis, or corresponding statistical mechanics cannot address is: what is the likelihood of learning the different solutions? This is the essential ambiguity of non-linear PCA – there are many solutions and the size of the basin of attractions of each is determined by the dynamics, not by local maxima of the nonlinear correlation measure.

As an example, we consider the partitioning of the neurons described in section 2. These neurons act much like neurons in competitive networks, they become tuned to individual patterns or highly correlated clusters. Given that the density of patterns in the input set is $\rho(\vec{x})$, what is the probability $p(\vec{x})$ that a neuron will become tuned to this pattern. It is often said that the desired result should be $p(\vec{x}) = \rho(\vec{x})$, although for Kohonen 1-d feature maps has been shown to be $p(\vec{x}) = \rho(\vec{x})^{2/3}$ (see for example, Hertz, Krogh, and Palmer 1991).

We have found that he partitioning cannot be calculated by finding the optima of the objective function. For example, in the case of weakly correlated patterns, the global maxima is the most likely pattern, whereas all of the patterns are local maxima. To determine the partitioning, the basin of attraction of each pattern must be computed. This could be different for different dynamics with the same fixed point structure.

In order to determine the partitioning, the dynamics must be understood. The details will be described elsewhere (Prügel-Bennett and Shapiro, 1994). For the case of weakly correlated patterns, a neuron will learn a pattern for which

$$\rho(\vec{x^p})(V_0^p)^{b-1} > \rho(\vec{x^q})(V_0^q)^{b-1} \qquad \forall q \neq p.$$

Here $V_0^p$ is the initial overlap (before learning) of the neuron's weights with the $p$th pattern. This defines the basin of attraction for each pattern.

In the large $P$ limit and for random patterns

$$p(\vec{x}) \approx \rho(\vec{x})^\alpha \tag{11}$$

where $\alpha \approx 2\log(P)/(b-1)$, $P$ is the number of patterns, and where $b$ is a parameter that controls the non-linearity of the neuron's response. If $b$ is chosen so that $\alpha = 1$,

then the probability of a neuron learning a pattern will be proportional to the frequency with which the pattern is presented.

## 5  CONCLUSIONS

The relationship between a non-linear generalization of Oja's rule and a non-linear generalization of PCA was reviewed. Non-linear PCA is equivalent to maximizing a objective function which is a statistical measure of the data set. The objective function optimized is determined by the form of the activation function of the neuron. Viewing the neuron in this way is useful, because rather than solving the dynamics, one can use methods of statistical mechanics or other methods to find the maxima of the objective function. Since this function has many local maxima, however, these techniques cannot determine how the solutions are partitioned amoung the neurons. To determine this, the dynamics must be solved.

### Acknowledgements

This work was supported by SERC grant GRG20912.

### References

J. Hertz, A. Krogh, and R.G. Palmer. (1991). Introduction to the Theory of Neural Computation. Addison-Wesley.

J. Karhunen and J. Joutsensalo. (1992) Nonlinear Hebbian algorithms for sinusoidal frequency estimation, in Artificial Neural Networks, 2, I. Akeksander and J. Taylor, editors, North-Holland.

Erkki Oja. (1982) A simplified neuron model as a principal component analyzer. em J. Math. Bio., 15:267-273.

Erkki Oja. (1989) Neural networks, principal components, and subspaces. *Int. J. of Neural Systems*, 1(1):61–68.

E. Oja, H. Ogawa, and J. Wangviwattan. (1992) Principal Component Analysis by homogeneous neural networks: Part II: analysis and extension of the learning algorithms *IEICE Trans. on Information and Systems, E75-D*, 3, pp 376–382.

E. Oja. (1993) Nonlinear PCA: algorithms and applications, in Proceedings of World Congress on Neural Networks, Portland, Or. 1993.

A. Prugel-Bennett and Jonathan L. Shapiro. (1993) Statistical Mechanics of Unsupervised Hebbian Learning. *J. Phys. A:* 26, 2343.

A. Prugel-Bennett and Jonathan L. Shapiro. (1994) The Partitioning Problem for Unsupervised Learning for Non-linear Neurons. *J. Phys. A* to appear.

T. D. Sanger. (1989) Optimal Unsupervised Learning in a Single-Layer Linear Feedforward Neural Network. *Neural Networks* 2, 459–473.

Jonathan L. Shapiro and A. Prugel-Bennett (1992), Unsupervised Hebbian Learning and the Shape of the Neuron Activation Function, in Artificial Neural Networks, 2, I. Akeksander and J. Taylor, editors, North-Holland.

W. Softky and D. Kammen (1991). Correlations in High Dimensional or Asymmetric Data Sets: Hebbian Neuronal Processing. *Neural Networks* 4, pp 337–347.

J. Taylor, (1993) Forms of Memory, in Proceedings of World Congress on Neural Networks, Portland, Or. 1993.